# Learning Discriminative Feature Transforms to Low Dimensions in Low Dimensions

**Kari Torkkola**
Motorola Labs, 7700 South River Parkway, MD ML28, Tempe AZ 85284, USA
*Kari.Torkkola@motorola.com    http://members.home.net/torkkola*

## Abstract

The marriage of Renyi entropy with Parzen density estimation has been shown to be a viable tool in learning discriminative feature transforms. However, it suffers from computational complexity proportional to the square of the number of samples in the training data. This sets a practical limit to using large databases. We suggest immediate divorce of the two methods and remarriage of Renyi entropy with a semi-parametric density estimation method, such as a Gaussian Mixture Models (GMM). This allows all of the computation to take place in the low dimensional target space, and it reduces computational complexity proportional to square of the number of components in the mixtures. Furthermore, a convenient extension to Hidden Markov Models as commonly used in speech recognition becomes possible.

## 1  Introduction

Feature selection or feature transforms are important aspects of any pattern recognition system. Optimal feature selection coupled with a particular classifier can be done by actually training and evaluating the classifier using all combinations of available features. Obviously this *wrapper* strategy does not allow learning feature transforms, because all possible transforms cannot be enumerated. Both feature selection and feature transforms can be learned by evaluating some criterion that reflects the "importance" of a feature or a number of features jointly. This is called the *filter* configuration in feature selection. An optimal criterion for this purpose would naturally reflect the Bayes error rate. Approximations can be used, for example, based on Bhattacharyya bound or on an interclass divergence criterion. These are usually accompanied by a parametric estimation, such as Gaussian, of the densities at hand [6, 12]. The classical Linear Discriminant Analysis (LDA) assumes all classes to be Gaussian with a shared single covariance matrix [5]. Heteroscedastic Discriminant Analysis (HDA) extends this by allowing each of the classes have their own covariances [9].

Maximizing a particular criterion, the joint mutual information (MI) between the features and the class labels [1, 17, 16, 13], can be shown to minimize the lower bound of the classification error [3, 10, 15]. However, MI according to the popular definition of Shannon can be computationally expensive. Evaluation of the joint MI of a number of variables is plausible through histograms, but only for a few variables [17]. As a remedy, Principe et al showed in [4, 11, 10] that using Renyi's entropy instead of Shannon's, combined with Parzen density estimation, leads to expressions of mutual information with computational complexity of $O(N^2)$, where $N$ is the number of samples in the training set. This method can be formulated to express the mutual information between continuous variables and discrete class labels in order to learn dimension-reducing feature transforms, both linear

[15] and non-linear [14], for pattern recognition. One must note that regarding finding the extrema, both definitions of entropy are equivalent (see [7] pages 118,406, and [8] page 325).

This formulation of MI evaluates the effect of each sample to every other sample in the transformed space through the Parzen density estimation kernel. This effect can also called as the "information force". Thus large/huge databases are hard to use due to the $O(N^2)$ complexity.

To remedy this problem, and also to alleviate the difficulties in Parzen density estimation in high-dimensional spaces ($d > 8$), we present a formulation combining the mutual information criterion based on Renyi entropy with a semi-parametric density estimation method using Gaussian Mixture Models (GMM). In essence, Parzen density estimation is replaced by GMMs. In order to evaluate the MI, evaluating mutual interactions between mixture components of the GMMs suffices, instead of having to evaluate interactions between all pairs of samples. An approach that maps an output space GMM back to input space and again to output space through the adaptive feature transform is taken. This allows all of the computation to take place in the target low dimensional space. Computational complexity is reduced proportional to the square of the number of components in the mixtures.

This paper is structured as follows. An introduction is given to the maximum mutual information (MMI) formulation for discriminative feature transforms using Renyi entropy and Parzen density estimation. We discuss different strategies to reduce its computational complexity, and we present a formulation based on GMMs. Empirical results are presented using a few well known databases, and we conclude by discussing a connection to Hidden Markov Models.

## 2 MMI for Discriminative Feature Transforms

Given a set of training data $\{x_i, c_i\}$ as samples of a continuous-valued random variable $X$, $x_i \in R^D$, and class labels as samples of a discrete-valued random variable $C$, $c_i \in \{1, 2, ..., N_c\}, i \in [1, N]$, the objective is to find a transformation (or its parameters $w$) to $y_i \in R^d, d < D$ such that $y_i = g(w, x_i)$ that maximizes $I(C, Y)$, the mutual information (MI) between transformed data $Y$ and class labels $C$. The procedure is depicted in Fig. 1. To this end, we need to express $I$ as a function of the data set, $I(\{y_i, c_i\})$, in a differentiable form. Once that is done, we can perform gradient ascent on $I$ as follows

$$w_{t+1} = w_t + \eta \frac{\partial I}{\partial w} = w_t + \eta \sum_{i=1}^{N} \frac{\partial I}{\partial y_i} \frac{\partial y_i}{\partial w}. \tag{1}$$

To derive an expression for MI using a non-parametric density estimation method we apply Renyi's quadratic entropy instead of Shannon's entropy as described in [10, 15] because of its computational advantages. Estimating the density $p(y)$ of $Y$ as a sum of spherical Gaussians each centered at a sample $y_i$, the expression of Renyi's quadratic entropy of $Y$ is

$$
\begin{aligned}
H_R(Y) &= -\log \int_y p(y)^2 dy \\
&= -\log \frac{1}{N^2} \int_y \left( \sum_{k=1}^{N} \sum_{j=1}^{N} G(y - y_k, \sigma^2 I) G(y - y_j, \sigma^2 I) \right) dy \\
&= -\log \frac{1}{N^2} \sum_{k=1}^{N} \sum_{j=1}^{N} G(y_k - y_j, 2\sigma^2 I).
\end{aligned}
\tag{2}
$$

Above, use is made of the fact that the convolution of two Gaussians is a Gaussian. Thus Renyi's quadratic entropy can be computed as a sum of local interactions as defined by the kernel, over all pairs of samples.

In order to use this convenient property, a measure of mutual information making use of quadratic functions of the densities would be desirable. Between a discrete variable $C$ and a continuous variable $Y$ such a measure has been derived in [10, 15] as follows:

$$I_T(C,Y) = \sum_c \int_{\boldsymbol{y}} p(c,\boldsymbol{y})^2 d\boldsymbol{y} + \sum_c \int_{\boldsymbol{y}} p(c)^2 p(\boldsymbol{y})^2 d\boldsymbol{y} - 2\sum_c \int_{\boldsymbol{y}} p(c,\boldsymbol{y})p(c)p(\boldsymbol{y}) d\boldsymbol{y} \quad (3)$$

We use $J_p$ for the number of samples in class $p$, $y_k$ for $k$th sample regardless of its class, and $y_{pj}$ for the same sample, but emphasizing that it belongs to class $p$, with index $j$ within the class. Expressing densities as their Parzen estimates with kernel width $\sigma$ results in

$$I_T(\{\boldsymbol{y}_i, c_i\}) = \frac{1}{N^2} \sum_{p=1}^{N_c} \sum_{k=1}^{J_p} \sum_{l=1}^{J_p} G(\boldsymbol{y}_{pk} - \boldsymbol{y}_{pl}, 2\sigma^2 I)$$

$$+ \frac{1}{N^2} \left( \sum_{p=1}^{N_c} \left( \frac{J_p}{N} \right)^2 \right) \sum_{k=1}^{N} \sum_{l=1}^{N} G(\boldsymbol{y}_k - \boldsymbol{y}_l, 2\sigma^2 I)$$

$$- 2\frac{1}{N^2} \sum_{p=1}^{N_c} \frac{J_p}{N} \sum_{j=1}^{J_p} \sum_{k=1}^{N} G(\boldsymbol{y}_{pj} - \boldsymbol{y}_k, 2\sigma^2 I) \quad (4)$$

Mutual information $I_T(\{\boldsymbol{y}_i, c_i\})$ can now be interpreted as an *information potential* induced by samples of data in different classes. It is now straightforward to derive partial $\partial I/\partial \boldsymbol{y}_i$ which can accordingly be interpreted as an *information force* that other samples exert to sample $\boldsymbol{y}_i$. The three components of the sum give rise to following three components of the information force: [1]Samples within the same class attract each other, [2]All samples regardless of class attract each other, and [3]Samples of different classes repel each other. This force, coupled with the latter factor $\partial \boldsymbol{y}_i/\partial \boldsymbol{w}$ inside the sum in (1), tends to change the transform in such a way that the samples in transformed space move into the direction of the information force, and thus increase the MI criterion $I(\{\boldsymbol{y}_i, c_i\})$. See [15] for details.

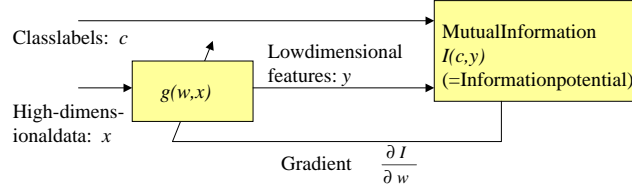

Figure 1: Learning feature transforms by maximizing the mutual information between class labels and transformed features.

Each term in (4) consists of a double sum of Gaussians evaluated using the pairwise distance between the samples. The first component consists of a sum of these interactions *within each class*, the second of all interactions *regardless of class*, and the third of a sum of the interactions of *each class against all other samples*. The bulk of computation consists of evaluating these $N^2/2$ Gaussians, and forming the sums of those. Information force, the gradient of $I_T$, makes use of the same Gaussians, in addition to pairwise differences of the samples [15]. For large $N$, complexity of $O(N^2)$ is a problem. Thus, the rest of the paper explores possibilities of reducing the computation to make the method applicable to large databases.

## 3   How to Reduce Computation?

In essence, we are trying to learn a transform that minimizes the class density overlap in the output space while trying to drive each class into a singularity. Since kernel density estimate results in a sum of kernels over samples, a divergence measure between the densities necessarily requires $O(N^2)$ operations. The only alternatives to reduce this complexity are either to reduce $N$, or to form simpler density estimates.

Two straightforward ways to achieve the former are clustering or random sampling. In this case clustering needs to be performed in the high-dimensional input space, which may be difficult and computationally expensive itself. A transform is then learned to find a representation that discriminates the cluster centers or the random samples belonging to different classes. Details of the densities may be lost, more so with random sampling, but at least this might bring the problem down to a computable level.

The latter alternative can be accomplished by a GMM, for example. A GMM is learned in the low-dimensional output space for each class, and now, instead of comparing samples against each other, comparing samples against the components of the GMMs suffices. However, as the parameters of the transform are being learned iteratively, the $\{y_i\}$ will change at each iteration, and the GMMs need to be estimated again. There is no guarantee that the change to the transform and to the $\{y_i\}$ is so small that simple re-estimation based on previous GMMs would suffice. However, this depends on the optimization method used.

A further step in reducing computation is to compare GMMs of different classes in the output space against each other, instead of comparing the actual samples. In addition to the inconvenience of re-estimation, we lack now the notion of "mapping". Nothing is being transformed by $g$ from the input space to the output space, such that we could change the transform in order to increase the MI criterion. Although it would be possible now to evaluate the effect of each sample to each mixture component, and the effect of each component to the MI, that is, $\frac{\partial I}{\partial \boldsymbol{w}} = \sum_i \sum_k \frac{\partial I}{\partial \boldsymbol{G}_i} \frac{\partial \boldsymbol{G}_i}{\partial \boldsymbol{y}_k} \frac{\partial \boldsymbol{y}_k}{\partial \boldsymbol{w}}$, due to the double summing, we will pursue the mapping strategy outlined in the following section.

## 4   Two GMM Mapping Strategies

**IO-mapping**. If the GMM is available in the high-dimensional input space, those models can be directly mapped into the output space by the transform. Let us call this case the *IO-mapping*. Writing the density of class $p$ as a GMM with $K_p$ mixture components and $h_{pj}$ as their mixture weights we get

$$p(\boldsymbol{x}|c_p) = \sum_{j=1}^{K_p} h_{pj} G(\boldsymbol{x} - \boldsymbol{\mu}_{pj}, \Sigma_{pj}) \tag{5}$$

We consider now only linear transforms. The transformed density in the low-dimensional output space is then simply

$$p(\boldsymbol{y}|c_p) = \sum_{j=1}^{K_p} h_{pj} G(\boldsymbol{y} - W\boldsymbol{\mu}_{pj}, W\Sigma_{pj}W^T) \tag{6}$$

Now, the mutual information in the output space between class labels and the densities as transformed GMMs can be expressed as a function of $W$, and it will be possible to evaluate $\partial I / \partial W$ to insert into (1). A great advantage of this strategy is that once the input space GMMs have been created (by the EM-algorithm, for example), the actual training data needs not be touched at all during optimization! This is thus a very viable approach if the GMMs are already available in the high-dimensional input space (see Section 7), or if it is not too expensive or impossible to estimate them using the EM-algorithm. However, this might not be the case.

**OIO-mapping**. An alternative is to construct a GMM model for the training data in the low-dimensional output space. Since getting there requires a transform, the GMM is constructed after having transformed the data using, for example, a random or an informed guess as the transform. Density estimated from the samples in the output space for class $p$ is

$$p(\boldsymbol{y}|c_p) = \sum_{j=1}^{K_p} h_{pj} G(\boldsymbol{y} - \boldsymbol{m}_{pj}, S_{pj}) \tag{7}$$

Once the output space GMM is constructed, the same samples are used to construct a GMM in the input space using the *same exact assignments of samples to mixture components* as the output space GMMs have. Running the EM-algorithm in the input space is now unnecessary since we know which samples belong to which mixture components. Similar strategy has been used to learn GMMs in high dimensional spaces [2]. Let us now use the notation of Eq.(5) to denote this density also in the input space. As a result, we have GMMs in both spaces and a transform mapping between the two. The transform can be learned as in the IO-mapping, by using the equalities $\boldsymbol{m}_{pj} = W\boldsymbol{\mu}_{pj}$ and $S_{pj} = W\Sigma_{pj}W^T$. This case will be called *OIO-mapping*. The biggest advantage is now avoiding to operate in the high-dimensional input space at all, not even the one time in the beginning of the procedure.

## 5 Learning the Transform through Mapped GMMs

We present now the derivation of adaptation equations for a linear transform that apply to either mapping. The first step is to express the MI as a function of the GMM that is constructed in the output space. This GMM is a function of the transform matrix $W$, through the mapping of the input space GMM to the output space GMM. The second step is to compute its gradient $\partial I/\partial W$ and to make use of it in the first half of Equation (1).

### 5.1 Information Potential as a Function of GMMs

GMM in the output space for each class is already expressed in (7). We need the following equalities: $p(c_p, \boldsymbol{y}) = P_p\, p(\boldsymbol{y}|c_p)$, where $P_p$ denotes the class prior, and $p(\boldsymbol{y}) = \sum_{p=1}^{N_c} p(c_p, \boldsymbol{y})$.

Let us denote the three terms in (3) as $V_{IN}$, $V_{ALL}$, and $-2V_{BTW}$. Then we have

$$V_{IN} = \sum_c \int_{\boldsymbol{y}} p(c, \boldsymbol{y})^2 d\boldsymbol{y} = \sum_{p=1}^{N_c} \int_{\boldsymbol{y}} P_p^2 \left( \sum_{i=1}^{K_p} h_{pi} G(\boldsymbol{y} - \boldsymbol{m}_{pi}, S_{pi}) \right)^2 d\boldsymbol{y}$$

$$= \sum_{p=1}^{N_c} P_p^2 \sum_{i=1}^{K_p} \sum_{j=1}^{K_p} h_{pi} h_{pj} G(\boldsymbol{m}_{pi} - \boldsymbol{m}_{pj}, S_{pi} + S_{pj}) \tag{8}$$

To compact the notation, we change the indexing, and make the substitutions $\boldsymbol{m}_{kl} = \boldsymbol{m}_k - \boldsymbol{m}_l$, $S_{kl} = S_k + S_l$, $G(k,l) = G(\boldsymbol{m}_{kl}, S_{kl})$, $V(k,l) = P_k P_l h_k h_l G(k,l)$, where $k, l \in [1...d_h]$, and $d_h$ is the total number of mixture components, and $V_{pq} = \sum_{k \in c_p} \sum_{l \in c_q} V(k,l)$. Now we can write $V_{IN}$, $V_{ALL}$, and $V_{BTW}$ in a convenient form.

$$V_{IN} = \sum_{p=1}^{N_c} V_{pp} \qquad V_{ALL} = \left(\sum_{r=1}^{N_c} P_r^2\right) \sum_{p=1}^{N_c} \sum_{q=1}^{N_c} V_{pq} \qquad V_{BTW} = \sum_{p=1}^{N_c} P_p \sum_{q=1}^{N_c} V_{pq} \tag{9}$$

### 5.2 Gradient of the Information Potential

As each Gaussian mixture component is now a function of the corresponding input space component and the transform matrix $W$, it is straightforward (albeit tedious) to write the gradient $\partial I_T / \partial W$. Since each of the three terms in $I_T$ is composed of different sums of $G(k, l)$, we need its gradient as

$$\frac{\partial}{\partial W} G(k, l) = \frac{\partial}{\partial W} G(\boldsymbol{m}_{kl}, S_{kl}) = \frac{\partial}{\partial W} G(W \boldsymbol{\mu}_{kl}, W \Sigma_{kl} W^T) \qquad (10)$$

where the input space GMM parameters are $\boldsymbol{\mu}_{kl} = \boldsymbol{\mu}_k - \boldsymbol{\mu}_l$ and $\Sigma_{kl} = \Sigma_k + \Sigma_l$ with the equalities $\boldsymbol{m}_{kl} = W \boldsymbol{\mu}_{kl}$ and $S_{kl} = W \Sigma_{kl} W^T$.

$G(k, l)$ expresses the convolution of two mixture components in the output space. As we also have those components in the high-dimensional input space, the gradient expresses how this convolution in the output space changes, as $W$ that maps the mixture components to the output space, is being changed. The mutual information measure is defined in terms of these convolutions, and maximizing it tends to find a $W$ that (crudely stated) minimizes these convolutions between classes and maximizes them within classes. The desired gradient of the Gaussian with respect to the transform matrix is as follows:

$$\frac{\partial}{\partial W} G(k, l) = -G(k, l) S_{kl}^{-1} \left[ \left( I - \boldsymbol{m}_{kl} \boldsymbol{m}_{kl}^T S_{kl}^{-1} \right) W \Sigma_{kl} + \boldsymbol{m}_{kl} \boldsymbol{\mu}_{kl}^T \right] \qquad (11)$$

The total gradient $\partial I_T / \partial W$ can now be obtained simply by replacing $G(k, l)$ in (8) and (9) by the above gradient.

In evaluating $I_T$, the bulk of computation is in evaluating the $V_{pq}$, the componentwise convolutions. Computational complexity is now $O(d_h^2)$. In addition, the $\partial I_T / \partial W$ requires pairwise sums and differences of the mixture parameters in the input space, but these need only be computed once.

## 6 Empirical Results

The first step in evaluating this approach is to compare its performance to the computationally more expensive MMI feature transforms that use Parzen density estimation. To this end, we repeated the pattern recognition experiments of [15] using exactly the same LVQ-classifier. These experiments were done using five publicly available databases that are very different in terms of the amount of data, dimension of data, and the number of training instances. For details of the data sets, please see [15]. OIO-mapping was used with 3-5 diagonal Gaussians per class to learn a dimension-reducing linear transform. Gradient ascent was used for optimization[1]. Results are presented in Tables 1 - 5. The last column denotes the original dimensionality of the data set.

As a figure of the overall performance, the average over all five databases and all reduced dimensions, which ranged from one up to the original dimension minus one, was 69.6% for PCA, 77.8% for the MMI-Parzen combination, and 77.0% for the MMI-GMM combination (30 tests altogether). For LDA this figure cannot be calculated since some databases had a small $N_c$ and LDA can only produce $N_c - 1$ features. The results are very satisfactory since the best we could hope for is performance equal to the MMI-Parzen combination. Thus a very significant reduction in computation caused only a minor drop in performance with this classifier.

## 7 Discussion

We have presented a method to learn discriminative feature transforms using Maximum Mutual Information as the criterion. Formulating MI using Renyi entropy, and Gaussian

Table 1: Accuracy on the Phoneme test data set using LVQ classifier.

| Output dimenson | 1 | 2 | 3 | 4 | 6 | 9 | 20 |
|---|---|---|---|---|---|---|---|
| PCA | 7.6 | 70.0 | 76.8 | 81.1 | 84.2 | 87.3 | 90.0 |
| LDA | 5.1 | 66.0 | 74.7 | 80.2 | 82.8 | 86.0 | - |
| MMI-Parzen | 15.5 | 68.5 | 75.2 | 80.2 | 82.6 | 85.3 | - |
| MMI-GMM | 21.4 | 70.4 | 76.8 | 80.2 | 82.6 | 87.7 | - |

Table 2: Accuracy on the Landsat test data set using LVQ classifier.

| Output dimension | 1 | 2 | 3 | 4 | 9 | 15 | 36 |
|---|---|---|---|---|---|---|---|
| PCA | 41.2 | 81.5 | 85.8 | 87.8 | 89.4 | 90.3 | 90.4 |
| LDA | 42.5 | 75.7 | 86.2 | 87.2 | 88.8 | 90.0 | - |
| MMI-Parzen | 65.1 | 82.0 | 86.4 | 86.2 | 87.6 | 89.5 | - |
| MMI-GMM | 65.0 | 80.4 | 86.1 | 88.3 | 87.4 | 89.1 | - |

Table 3: Accuracy on the Letter test data set using LVQ classifier.

| Output dimension | 1 | 2 | 3 | 4 | 6 | 8 | 16 |
|---|---|---|---|---|---|---|---|
| PCA | 4.5 | 16.0 | 36.0 | 53.2 | 75.2 | 82.5 | 92.4 |
| LDA | 13.4 | 38.0 | 53.1 | 68.1 | 80.3 | 86.3 | - |
| MMI-Parzen | 16.4 | 50.3 | 62.8 | 70.9 | 82.4 | 88.6 | - |
| MMI-GMM | 15.7 | 42.4 | 48.3 | 68.5 | 80.9 | 86.6 | - |

Table 4: Accuracy on the Pipeline data set using LVQ classifier.

| Output dimension | 1 | 2 | 3 | 4 | 5 | 7 | 12 |
|---|---|---|---|---|---|---|---|
| PCA | 41.5 | 88.0 | 87.8 | 89.7 | 96.4 | 97.2 | 99.0 |
| LDA | 98.4 | 98.8 | - | - | - | - | - |
| MMI-Parzen | 99.4 | 99.1 | 98.9 | 99.2 | 98.9 | 99.0 | - |
| MMI-GMM | 91.3 | 98.8 | 99.1 | 98.9 | 99.1 | 98.7 | - |

Table 5: Accuracy on the Pima data set using LVQ classifier.

| Output dimension | 1 | 2 | 3 | 4 | 5 | 6 | 8 |
|---|---|---|---|---|---|---|---|
| PCA | 64.4 | 73.0 | 75.2 | 74.1 | 75.6 | 74.7 | 74.7 |
| LDA | 65.8 | - | - | - | - | - | - |
| MMI-Parzen | 72.0 | 77.5 | 78.7 | 78.5 | 78.3 | 78.3 | - |
| MMI-GMM | 73.9 | 79.7 | 79.4 | 77.9 | 76.7 | 77.5 | - |

Mixture Models as a semi-parametric density estimation method, allows all of the computation to take place in the low-dimensional transform space. Compared to previous formulation using Parzen density estimation, large databases become now a possibility.

A convenient extension to Hidden Markov Models (HMM) as commonly used in speech recognition becomes also possible. Given an HMM-based speech recognition system, the state discrimination can be enhanced by learning a linear transform from some high-dimensional collection of features to a convenient dimension. Existing HMMs can be converted to these high-dimensional features using so called single-pass retraining (compute all probabilities using current features, but do re-estimation using a the high-dimensional set of features). Now a state-discriminative transform to a lower dimension can be learned using the method presented in this paper. Another round of single-pass retraining then converts existing HMMs to new discriminative features.

A further advantage of the method in speech recognition is that the state separation in the transformed output space is measured in terms of the separability of the data represented as Gaussian mixtures, not in terms of the data itself (actual samples). This should be advantageous regarding recognition accuracies since HMMs have the same exact structure.

## Footnotes

[1]Example video clips can be viewed at http://members.home.net/torkkola/mmi.

# References

[1] R. Battiti. Using mutual information for selecting features in supervised neural net learning. *Neural Networks*, 5(4):537–550, July 1994.

[2] Sanjoy Dasgupta. Experiments with random projection. In *Proceedings of the 16th Conference on Uncertainty in Artificial Intelligence*, pages 143–151, Stanford, CA, June30 - July 3 2000.

[3] R.M. Fano. *Transmission of Information: A Statistical theory of Communications*. Wiley, New York, 1961.

[4] J.W. Fisher III and J.C. Principe. A methodology for information theoretic feature extraction. In *Proc. of IEEE World Congress On Computational Intelligence*, pages 1712–1716, Anchorage, Alaska, May 4-9 1998.

[5] K. Fukunaga. *Introduction to statistical pattern recognition (2nd edition)*. Academic Press, New York, 1990.

[6] Xuan Guorong, Chai Peiqi, and Wu Minhui. Bhattacharyya distance feature selection. In *Proceedings of the 13th International Conference on Pattern Recognition*, volume 2, pages 195 – 199. IEEE, 25-29 Aug. 1996.

[7] J.N. Kapur. *Measures of information and their applications*. Wiley, New Delhi, India, 1994.

[8] J.N. Kapur and H.K. Kesavan. *Entropy optimization principles with applications*. Academic Press, San Diego, London, 1992.

[9] Nagendra Kumar and Andreas G. Andreou. Heteroscedastic discriminant analysis and reduced rank HMMs for improved speech recognition. *Speech Communication*, 26(4):283–297, 1998.

[10] J.C. Principe, J.W. Fisher III, and D. Xu. Information theoretic learning. In Simon Haykin, editor, *Unsupervised Adaptive Filtering*. Wiley, New York, NY, 2000.

[11] J.C. Principe, D. Xu, and J.W. Fisher III. Pose estimation in SAR using an information-theoretic criterion. In *Proc. SPIE98*, 1998.

[12] George Saon and Mukund Padmanabhan. Minimum bayes error feature selection for continuous speech recognition. In Todd K. Leen, Thomas G. Dietterich, and Volker Tresp, editors, *Advances in Neural Information Processing Systems 13*, pages 800–806. MIT Press, 2001.

[13] Janne Sinkkonen and Samuel Kaski. Clustering based on conditional distributions in an auxiliary space. *Neural Computation*, 14:217–239, 2002.

[14] Kari Torkkola. Nonlinear feature transforms using maximum mutual information. In *Proceedings of the IJCNN*, pages 2756–2761, Washington DC, USA, July 15-19 2001.

[15] Kari Torkkola and William Campbell. Mutual information in learning feature transformations. In *Proceedings of the 17th International Conference on Machine Learning*, pages 1015–1022, Stanford, CA, USA, June 29 - July 2 2000.

[16] N. Vlassis, Y. Motomura, and B. Krose. Supervised dimension reduction of intrinsically low-dimensional data. *Neural Computation*, 14(1), January 2002.

[17] H. Yang and J. Moody. Feature selection based on joint mutual information. In *Proceedings of International ICSC Symposium on Advances in Intelligent Data Analysis*, Rochester, New York, June 22-25 1999.
